# Sampling Methods for Unsupervised Learning

**Rob Fergus**[*] **& Andrew Zisserman**
Dept. of Engineering Science
University of Oxford
Parks Road, Oxford OX1 3PJ, UK.
{fergus,az }@robots.ox.ac.uk

**Pietro Perona**
Dept. Electrical Engineering
California Institute of Technology
Pasadena, CA 91125, USA.
perona@vision.caltech.edu

## Abstract

We present an algorithm to overcome the local maxima problem in estimating the parameters of mixture models. It combines existing approaches from both EM and a robust fitting algorithm, RANSAC, to give a data-driven stochastic learning scheme. Minimal subsets of data points, sufficient to constrain the parameters of the model, are drawn from proposal densities to discover new regions of high likelihood. The proposal densities are learnt using EM and bias the sampling toward promising solutions. The algorithm is computationally efficient, as well as effective at escaping from local maxima. We compare it with alternative methods, including EM and RANSAC, on both challenging synthetic data and the computer vision problem of alpha-matting.

## 1 Introduction

In many real world applications we wish to learn from data which is not labeled, to find clusters or some structure within the data. For example in Fig. 1(a) we have some clumps of data that are embedded in noise. Our goal is to automatically find and model them. Since our data has many components so must our model. Consequently the model will have many parameters and finding the optimal settings for these is a difficult problem. Additionally, in real world problems, the signal we are trying to learn is usually mixed in with a lot of irrelevant noise, as demonstrated by the example in Fig. 1(b). The challenge here is to find these lines reliably despite them only constituting a small portion of the data.

Images from Google, shown in Fig. 1(c), are typical of real world data, presenting both the challenges highlighted above. Our motivating real-world problem is to learn a visual model from the set of images returned by Google's image search on an object type (such as "camel", "tiger" or "bottles"), like those shown. Since text-based cues alone were used to compile the images, typically only 20%-50% images are visually consistent and the remainder may not even be images of the sought object type, resulting in a challenging learning problem.

Latent variable models provide a framework for tackling such problems. The parameters of these may be estimated using algorithms based on EM [2] in a maximum likelihood framework. While EM provides an efficient estimation scheme, it has a serious problem in that for complex models, a local maxima of the likelihood function is often reached rather than the global maxima. Attempts to remedy this problem include: annealed versions of EM [8]; Markov-Chain Monte-Carlo (MCMC) based clustering [4] and Split and Merge EM (SMEM) [9].

---

[*]corresponding author

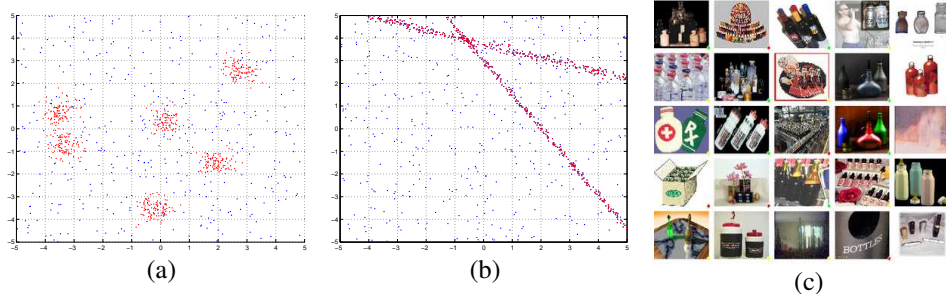

<center>(a)&emsp;&emsp;&emsp;&emsp;&emsp;&emsp;&emsp;&emsp;&emsp;&emsp;(b)&emsp;&emsp;&emsp;&emsp;&emsp;&emsp;&emsp;&emsp;&emsp;(c)</center>

Figure 1: The objective is to learn from contaminated data such as these: **(a)** Synthetic Gaussian data containing many components. **(b)** Synthetic line data with few components but with a large portion of background noise. **(c)** Images obtained by typing "bottles" into Google's image search.

Alternative approaches to unsupervised learning include the RANSAC [3, 5] algorithm and its many derivatives. These rely on stochastic methods and have proven highly effective at solving certain problems in Computer Vision, such as structure from motion, where the signal-to-noise ratios are typically very small.

In this paper we introduce an unsupervised learning algorithm that is based on both latent variable models and RANSAC-style algorithms. While stochastic in nature, it operates in data space rather than parameter space, giving a far more efficient algorithm than traditional MCMC methods.

## 2  Specification of the problem

We have a set of data $\mathbf{x} = \{\mathbf{x}_1 \dots \mathbf{x}_N\}$ with unseen labels $y = \{y_1 \dots y_N\}$ and a parametric mixture model with parameters $\boldsymbol{\theta}$, of the form:

$$p(\mathbf{x}|\boldsymbol{\theta}) = \sum_y p(\mathbf{x}, y|\boldsymbol{\theta}) = \sum_y p(\mathbf{x}|y, \boldsymbol{\theta})\, P(y|\boldsymbol{\theta}) \tag{1}$$

We assume the number of mixture components is known and equal to $C$. We also assume that the parametric form of the mixture components is given. One of these components will model the background noise, while the remainder fit the signal within the data. Thus the task is to find the value of $\boldsymbol{\theta}$ that maximizes the likelihood, $p(\mathbf{x}|\boldsymbol{\theta})$ of the data. This is not a straightforward as the dimensionality of $\boldsymbol{\theta}$ is large and the likelihood function is highly non-linear. Algorithms such as EM often get stuck in local maxima such as those illustrated in Fig. 2, and since they use gradient-descent alone, are unable to escape.

Before describing our algorithm, we first review the robust fitting algorithm RANSAC, from which we borrow several key concepts to enable us to escape from local maxima.

### 2.1  RANSAC

RANSAC (RANdom Sampling And Consensus) attempts to find global maxima by drawing random subset of points, fitting a model to them and then measuring their support from the data. A variant, MLESAC [7], gives a probabilistic interpretation of the original scheme which we now explain.

The basic idea is to draw at random and without replacement from $\mathbf{x}$, a set of $P$ samples for each of the $C$ components in our model; $P$ being the smallest number required to compute the parameters $\theta_c$ for each component. Let draw $i$ be represented by $\mathbf{z}^i$, a vector of length $N$ containing exactly $P$ ones, indicating the points selected with the rest being zeros. Thus $\mathbf{x}(\mathbf{z}^i)$ is the subset of points drawn from $\mathbf{x}$. From $\mathbf{x}(\mathbf{z}^i)$ we then compute the parameters for the component, $\theta_c^i$. Having done this for all components, we then estimate

the component mixing portions, $\boldsymbol{\pi}$ using EM (keeping the other parameters fixed), giving a set of parameters for draw $i$, $\boldsymbol{\theta}^i = \{\boldsymbol{\pi}, \theta_1^i \ldots \theta_C^i\}$. Using these parameters, we compute the likelihood over all the data: $p(\mathbf{x}|\boldsymbol{\theta}^i)$.

The entire process is repeated until either we exceed our maximum limit on the number of draws or we reach a pre-defined performance level. The final set of parameters are those that gave the highest likelihood: $\boldsymbol{\theta}^* = arg\,max_i\, p(\mathbf{x}|\boldsymbol{\theta}_i)$. Since this process explores a finite set of points in the space of $\boldsymbol{\theta}$, it is unlikely that the globally optimal point, $\boldsymbol{\theta}^{ML}$, will be found, but $\boldsymbol{\theta}^*$ should be close so that running EM from it is guaranteed to find the global optimum.

However, it is clear that the approach of sampling randomly, while guaranteed to eventually find a point close to $\boldsymbol{\theta}^{ML}$, is very inefficient since the number of possible draws scales exponentially with both $P$ and $C$. Hence it is only suitable for small values of both $P$ and $C$. While Tordoff *et. al.* [6] proposed drawing the samples from a non-uniform density, this approach involved incorporating auxiliary information about each sample point which may not be available for more general problems. However, Matas *et. al.* [1] propose general scheme to draw samples selectively from points tentatively classified as signal. This increases the efficiency of the sampling and motivates our approach.

## 3   Our approach – PROPOSAL

Our approach, which we name PROPOSAL (PROPOsal based SAmple Learning), combines aspects of both EM and RANSAC to produce a method with the robustness of RANSAC but with a far greater efficiency, enabling it to work on more complex models. The problem with RANSAC is that points are drawn randomly. Even after a large number of draws this random sampling continues, despite the fact that we may have already discovered a good, albeit local, maximum in our likelihood function.

The key idea in PROPOSAL is to draw samples from a *proposal density*. Initially this density is uniform, as in RANSAC, but as regions of high likelihood are discovered, we update it so that it gives a strong bias toward producing good draws again, increasing the efficiency of the sampling process. However, having found local maxima, we must still be able to escape and find the global maxima.

Local maxima are characterized by too many components in one part of the space and too few in another. To resolve this we borrow ideas from Split and Merge EM (SMEM) [9]. SMEM uses two types of discrete moves to discover superior maxima. In the first, a component in an underpopulated region is split into two new ones, while in the second two components in an overpopulated area are merged. These two moves are performed together to keep the number of components constant. For the local maxima encountered in Fig. 2(a), merging the green and blue components, while splitting the red component will yield a superior solution.

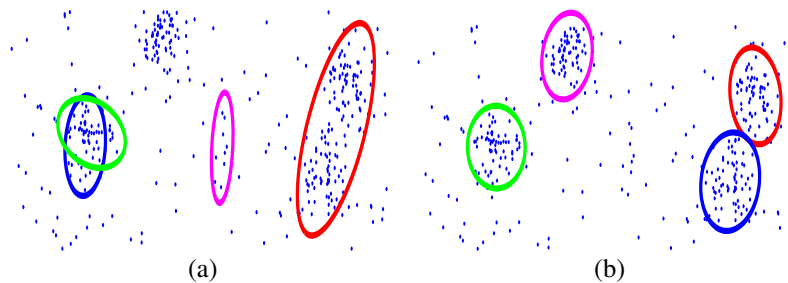

(a)                                                         (b)

Figure 2: **(a)** Examples of different types of local maxima encountered. The green and blue components on the left are overpopulating a small clump of data. The magenta component in the center models noise, while missing a clump altogether. The single red component on the right is inadequately modeling two clumps of data. **(b)** The global optimum solution.

PROPOSAL acts in a similar manner, by first finding components that are superfluous via two tests (described in section 3.3): (i) the *Evaporation* test – which would find the magenta component in Fig. 2(a) and (ii) the *Overlap* test – which would identify one of the green and blue components in Fig. 2(a). Then their proposal densities are adjusted so that they focus on data that is underpopulated by the model, thus subsequent samples are likely to discover a superior solution. An overview of the algorithm is as follows:

---

**Algorithm 1** PROPOSAL

---

**Require:** Data $\mathbf{x}$; Parameters: $C$, $\pi_{\min}$, $\epsilon$
  **for** $i = 1$ to $I^{\text{Max}}$ **do**
    **repeat**
        • For each component, $c$, compute parameters $\theta_c^i$ from $P$ points drawn from the proposal density $q_c(\mathbf{x}|\theta_c)$.
        • Estimate mixing portions, $\pi^i$, using EM, keeping $\theta_c^i$ fixed.
        • Compute the likelihood $L^i = \prod_n p(\mathbf{x}_n|\pi^i, \theta_1^i \dots \theta_C^i)$.
    **until** $L^i > L_{\text{Rough}}^{\text{Best}}$
    • Refine $\boldsymbol{\theta}^i$ using EM to give $\boldsymbol{\theta}^*$ with likelihood $L^*$.
    **if** $L^* > L^{\text{Best}}$ **then**
        • Update the proposal densities, $q(\mathbf{x}|\boldsymbol{\theta})$, using $\boldsymbol{\theta}^*$.
        • Apply the *Evaporation* and *Overlap* tests (using parameters $\pi_{\min}$ and $\epsilon$).
        • Reassign the proposal densities of any components failing the above tests.
        • Let $L_{\text{Rough}}^{\text{Best}} = L^i$; let $L^{\text{Best}} = L^*$ and let $\boldsymbol{\theta}^{\text{Best}} = \boldsymbol{\theta}^*$.
    **end if**
  **end for**
  Output: $\boldsymbol{\theta}^{\text{Best}}$ and $L^{\text{Best}}$.

---

We now elaborate on the various stages of the algorithm, using Fig. 3 as an example.

### 3.1 Sampling from data proposal densities

Each component, $c$, draws its samples from a proposal density, which is an empirical distribution of the form:

$$q_c(\mathbf{x}|\boldsymbol{\theta}) = \frac{\sum_{n=1}^N \delta(\mathbf{x} - \mathbf{x}_n)P(y = c|\mathbf{x}_n, \theta_c)}{\sum_{n=1}^N P(y = c|\mathbf{x}_n, \theta_c)} \tag{2}$$

where $P(y|\mathbf{x}, \boldsymbol{\theta})$ is the posterior on the labels:

$$P(y|\mathbf{x}, \boldsymbol{\theta}) = \frac{p(\mathbf{x}|y, \boldsymbol{\theta})P(y|\boldsymbol{\theta})}{\sum_y p(\mathbf{x}|y, \boldsymbol{\theta})P(y|\boldsymbol{\theta})} \tag{3}$$

Initially, $q(\mathbf{x}|\boldsymbol{\theta})$ is uniform, so we are drawing the points completely at random, but $q(\mathbf{x}|\boldsymbol{\theta})$ will become more peaked, biasing our draws toward the data picked out by the component, demonstrated in Fig. 3(c), which shows the non-uniform proposal densities for each component on a simulated problem. Note that if we are sampling *with* replacement, then $E[\mathbf{z}] = P(y|\mathbf{x}, \boldsymbol{\theta})$[1]. However, since we must avoid degenerate combinations of points, certain values of $\mathbf{z}$ are not permissible, so $E[\mathbf{z}] \to P(y|\mathbf{x}, \boldsymbol{\theta})$ as $N \to \infty$.

### 3.2 Computing model parameters

Each component $c$ has a subset of points picked out by $\mathbf{z}$ from which its parameters $\theta_c^i$ are estimated. Since each subset is of the minimal size required to constrain all parameters, this process is straightforward since it is usually closed-form. For the Gaussian example

in Fig. 3, we draw 3 points for each of the 4 Gaussian components, whose mean and covariance matrices are directly computed, using appropriate normalizations to give unbiased estimators of the population parameters.

Given $\theta_c^i$ for each component, the only unknown parameter is their relative weighting, $\boldsymbol{\pi} = P(y|\boldsymbol{\theta})$. This is estimated using EM. The E-step involves computing $P(y|\mathbf{x}, \boldsymbol{\theta})$ from (3). This can done efficiently since the component parameters are fixed, allowing the precomputation of $p(\mathbf{x}|y, \boldsymbol{\theta})$. The M-step is then $\pi_c = \frac{1}{N} \sum_{n=1}^{N} P(y = c|\mathbf{x}, \boldsymbol{\theta})$.

### 3.3 Updating proposal densities

Having obtained a rough model for draw $i$ with parameters $\boldsymbol{\theta}^i$ and likelihood $L^i$, we first see if its likelihood exceeds the likelihood of the previous best rough model, $L_{\text{Rough}}^{\text{Best}}$. If this is the case we refine the rough model to ensure that we are at an actual maximum since the sampling process limits us to a set of discrete points in $\boldsymbol{\theta}$-space, which are unlikely to be maxima themselves. Running EM again, this time updating all parameters and using $\boldsymbol{\theta}^i$ as an initialization, the parameters converge to $\boldsymbol{\theta}^*$, having likelihood $L^*$. If $L^*$ exceeds a second threshold (the previous best refined model's likelihood) $L^{\text{Best}}$, then we we recompute the proposal densities, as given in (2), using $P(y|\mathbf{x}, \boldsymbol{\theta}^*)$. The two thresholds are needed to avoid wasting time refining $\boldsymbol{\theta}^i$'s that are not initially promising. In updating the proposal densities, two tests are applied to $\boldsymbol{\theta}^*$:

1. *Evaporation test*: If $\pi_c < \pi_{\min}$, then the component is deemed to model noise, so is flagged for resetting. Fig. 3 illustrates this test.

2. *Overlap test*[2]: If for any two components, $a$ and $b$, $\frac{\|\theta_a^i - \theta_b^i\|^2}{\|\theta_a^i\|\,\|\theta_b^i\|} < \epsilon^2$, then the two components are judged to be fitting the same data. Component $a$ or $b$ is picked at random and flagged for resetting.

### 3.4 Resetting a proposal density

If a component's proposal density is to be reset, it is given a new density that maximizes the entropy of the mean proposal density $q_M(\mathbf{x}|\boldsymbol{\theta}) = \frac{1}{C} \sum_{c=1}^{C} q_c(\mathbf{x}|\boldsymbol{\theta})$.

By maximizing the entropy of $q_M(\mathbf{x}|\boldsymbol{\theta})$, we are ensuring that the samples will subsequently be drawn as widely as possible, maximizing the chances of escaping from the local minima. If $q_d(\mathbf{x}|\boldsymbol{\theta})$ are the proposal densities to be reset, then we wish to maximize:

$$H[q_M(\mathbf{x}|\boldsymbol{\theta})] = H\left[\frac{1}{D}\sum_d q_d(\mathbf{x}|\boldsymbol{\theta}) + \frac{1}{C-D}\sum_{c \neq d} q_d(\mathbf{x}|\boldsymbol{\theta})\right] \tag{4}$$

with the constraints that $\sum_n q_d(\mathbf{x}_n|\boldsymbol{\theta}) = 1 \,\forall\, d$ and $q_d(\mathbf{x}_n|\boldsymbol{\theta}) \geq 0 \,\forall\, n, d$. For brevity, let us define: $q_f(\mathbf{x}|\boldsymbol{\theta}) = \frac{1}{C-D}\sum_{c \neq d} q_d(\mathbf{x}|\boldsymbol{\theta})$.

Since a uniform distribution has the highest entropy, but $q_d(\mathbf{x}|\boldsymbol{\theta})$ cannot be negative, the optimal choice of $q_d(\mathbf{x}|\boldsymbol{\theta})$ will be zero everywhere, except for $\mathbf{x}$ corresponding to the smallest $k$ values of $q_f(\mathbf{x}|\boldsymbol{\theta})$. At these points $q_d(\mathbf{x}|\boldsymbol{\theta})$ must add to $q_f(\mathbf{x}|\boldsymbol{\theta})$ to give a constant $q_M(\mathbf{x}|\boldsymbol{\theta})$. We solve for $k$ using the other constraint, that probability mass of exactly $D/C$ must be added.

Thus $q_d(\mathbf{x}|\boldsymbol{\theta})$ be large where $q_f(\mathbf{x}|\boldsymbol{\theta})$ is small, giving the appealing result that the new component will draw preferentially from underpopulated portion of the data, as demonstrated in Fig. 3(d).

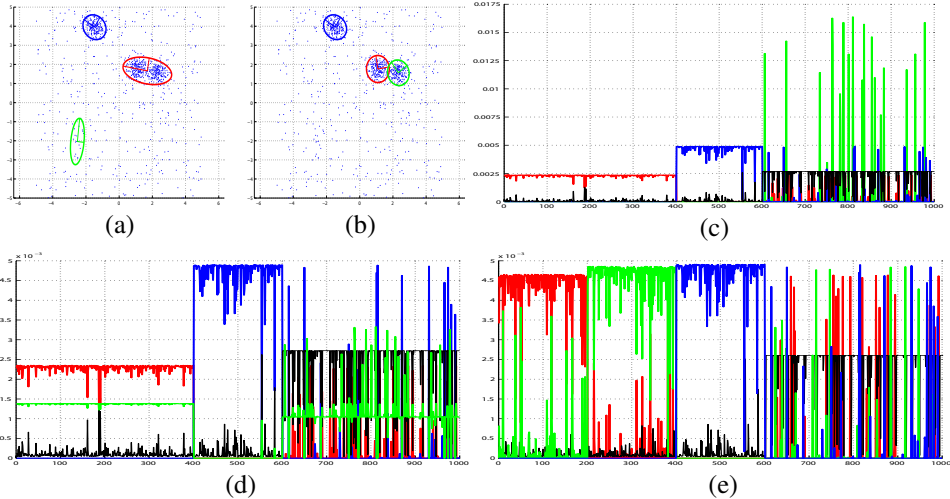

Figure 3: The *Evaporation* step in action. A local maximum is found in **(a)**. **(c)** shows the corresponding proposal densities for each component (black is the background model). Note how spiky the green density is, since it is only modeling a few data points. Since $\pi_{\text{green}} < \pi_{\text{min}}$, its proposal density is set to $q_d(\mathbf{x}|\boldsymbol{\theta})$, as shown in **(d)**. Note how $q_d(\mathbf{x}|\boldsymbol{\theta})$ is higher in the areas occupied by the red component which is a poor fit for two clumps of data. **(b)** The global maxima along with its proposal density **(e)**. Note that the data points are ordered for ease of visualization only.

## 4 Experiments

### 4.1 Synthetic experiments

We tested PROPOSAL on two types of synthetic data – mixtures of 2-D lines and Gaussians with uniform background noise. We compared six algorithms: Plain EM; Deterministic Annealing EM (DAEM)[8]; Stochastic EM (SEM)[10]; Split and Merge EM (SMEM); MLESAC and PROPOSAL. Four experiments were performed: two using lines and two with Gaussians. The first pair of experiments examined how many components the different algorithms could handle reliably. The second pair tested the robustness to background noise. In the Gaussian experiments, the model consisted of a mixture of 2-D Gaussian densities and a uniform background component. In the line experiments, the model consisted of a mixture of densities modeling the residual to the line with a Gaussian noise model, having a variance $\sigma$ that was also learnt. Each line component has therefore three parameters – its gradient; y-intercept and variance.

Each experiment was repeated 250 times with a different, randomly generated dataset, examples of which can be seen in Fig. 1(a) & (b). In each experiment, the same time was allocated for each algorithm, so for example, EM which ran quickly was repeated until it had spent the same amount of time as the slowest (usually PROPOSAL or SMEM), and the best result from the repeated runs taken. For simplicity, the *Overlap* test compared only the means of the distributions. Parameter values used for PROPOSAL were: $I = 200$, $\pi_{\text{min}} = 0.01$ and $\epsilon = 0.1$.

In the first pair of experiments, the number of components was varied from 2 upto 10 for lines and 20 for Gaussians. The background noise was held constant at 20%. The results are shown in Fig. 4. PROPOSAL clearly outperforms the other approaches. In the second pair of experiments, $C = 3$ components were used, with the background noise varying from 1% up to 99% . Parameters used were the same as for the first experiment. The results can be seen in Fig. 5. Both SMEM and PROPOSAL outperformed EM convincingly. PROPOSAL performed well down to 30% in the line case (i.e. 10% per line) and 20% in the Gaussian case.

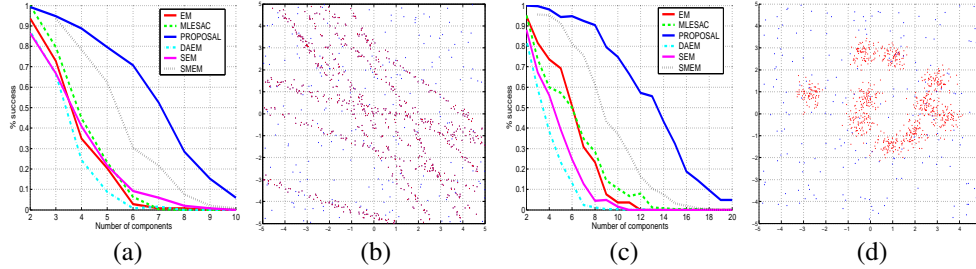

Figure 4: Experiments showing the robustness to the number of components in the model. The $x$-axis is the number of components ranging from 2 upwards. The $y$-axis is portion of correct solutions found from 250 runs, each having with a different randomly generated dataset. Key: EM (red solid); DAEM (cyan dot-dashed); SEM (magenta solid); SMEM (black dotted); MLESAC (green dashed) and PROPOSAL (blue solid). **(a)** Results for line data. **(b)** A typical line dataset for $C = 10$. **(c)** Results for Gaussian data. PROPOSAL is still achieving $75\%$ correct with 10 components - twice the performance of the next best algorithm (SMEM). **(d)** A typical Gaussian dataset for $C = 10$.

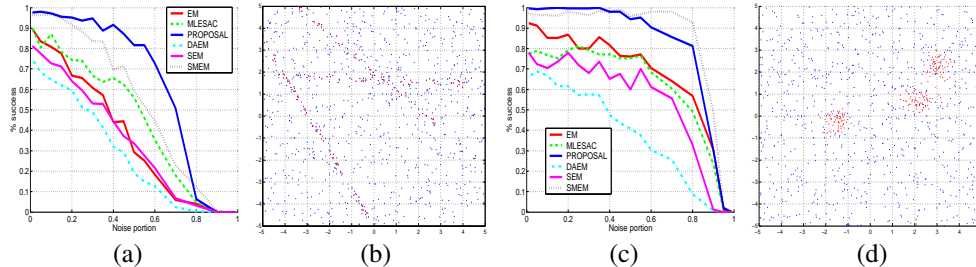

Figure 5: Experiments showing the robustness to background noise. The $x$-axis is the portion of noise, varying between $1\%$ and $99\%$. The $y$-axis is portion of correct solutions found. Key: EM (red solid); DAEM (cyan dot-dashed); SEM (magenta solid); SMEM (black dotted); MLESAC (green dashed) and PROPOSAL (blue solid). **(a)** Results for three component line data. **(b)** A typical line dataset for $80\%$ noise. **(c)** Results for three component Gaussian data. SMEM is marginally superior to PROPOSAL. **(d)** A typical Gaussian dataset for $80\%$ noise.

## 4.2 Real data experiments

We test PROPOSAL against other clustering methods on the computer vision problem of alpha-matting (the extraction of a foreground element from a background image by estimating the opacity for each pixel of the foreground element, see Figure 6 for examples). The simple approach we adopt is to first form a tri-mask (the composite image is divided into 3 regions: pixels that are definitely foreground; pixels that are definitely background and uncertain pixels). Two color models are constructed by clustering with a mixture of Gaussians the foreground and background pixels respectively. The opacity (alpha values) of the uncertain pixels are then determined by using comparing the color of the pixel under the foreground and background color models. Figure 7 compares the likelihood of the foreground and background color models clustered using EM, SMEM and PROPOSAL on two sets of images (11 face images and 5 dog images, examples of which are shown in Fig. 6). Each model is clustering $\sim 2 \times 10^4$ pixels in a 4-D space (R,G,B and edge strength) with a 10 component model. In the majority of cases, PROPOSAL can be seen to outperform SMEM which in turn out performs plain EM.

## 5 Discussion

In contrast to SMEM, MCEM [10] and MCMC [4], which operate in $\theta$-space, PROPOSAL is a *data-driven* approach. It prevalently examines the small portion of $\theta$-space which has support from the data. This gives the algorithm its robustness and efficiency. We have shown PROPOSAL to work well on synthetic data, outperforming many standard algorithms. On real data, PROPOSAL also convincingly beats SMEM and EM. One problem

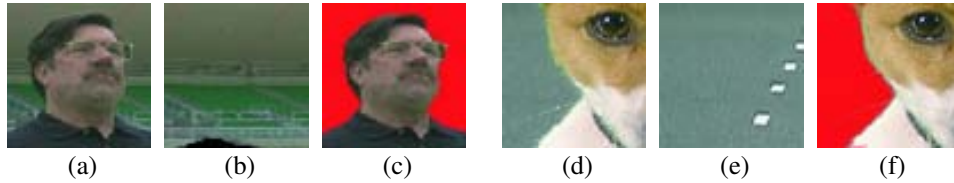

|   (a)   |   (b)   |   (c)   |   (d)   |   (e)   |   (f)   |

Figure 6: The alpha-matte problem. (a) & (d): Composite images. (b) & (e): Background images. (c) & (f): Desired object segmentation. This figure is best viewed in color.

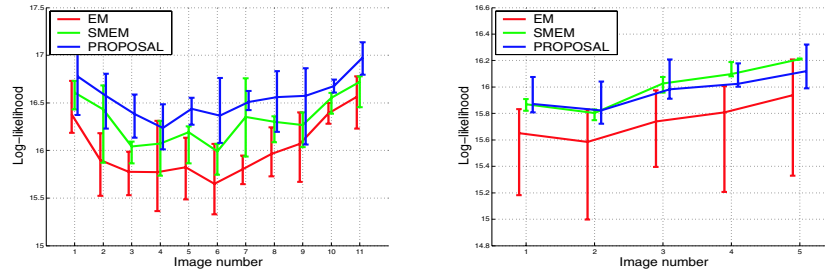

Figure 7: Clustering performance on (Left) 11 face images (e.g. Fig. 6(a)) and (Right) 5 dog images (e.g. Fig. 6(d)). $x$-axis is image number. $y$-axis is log-likelihood of foreground color model on foreground pixels plus log-likelihood of background color model on background pixels. Three clustering methods are shown: EM (red); SMEM (green) and PROPOSAL (blue). Line indicates mean of 10 runs from different random initializations while error bars show the best and worst models found from the 10 runs.

with PROPOSAL is that $P$ scales with the square of the dimension of the data (due to the number of terms in the covariance matrix) meaning for high dimensions, a very large number of draws would be needed to find new portions of data. Hence PROPOSAL is suited to problems of low dimension.

**Acknowledgments:** Funding was provided by EC Project CogViSys, EC NOE Pascal, Caltech CNSE, the NSF and the UK EPSRC. Thanks to F. Schaffalitzky & P. Torr for useful discussions.

## Footnotes

[1]Recall that $\mathbf{z}$ is a vector representing a draw of $P$ points from $q(\mathbf{x}|\theta)$. It is of length $N$ with exactly $P$ ones, the remaining elements being zero.

[2] An alternative overlap test would compare the responsibilities of each pair of components, $a$ and $b$: $\frac{P(y=a|\mathbf{x},\theta_a^i)^T P(y=b|\mathbf{x},\theta_b^i)}{\|P(y=a|\mathbf{x},\theta_a^i)\|\,\|P(y=b|\mathbf{x},\theta_b^i)\|} < \epsilon^2$.

# References

[1] Ondřej Chum, Jiří Matas, and Josef Kittler. Locally optimized ransac. In *DAGM 2003: Proceedings of the 25th DAGM Symposium*, pages 236–243, 2003.

[2] A. Dempster, N. Laird, and D. Rubin. Maximum likelihood from incomplete data via the em algorithm. *Journal of the Royal Statistical Society*, 39:1–38, 1976.

[3] M. A. Fischler and R. C. Bolles. Random sample consensus: A paradigm for model fitting with applications to image analysis and automated cartography. *Comm. ACM*, 24(6):381–395, 1981.

[4] S. Richardson and P.J. Green. On bayesian analysis of mixtures with an unknown number of components. *Journal of the Royal Statistical Society*, 59(4):731–792, 1997.

[5] C.V. Stewart. Robust parameter estimation. *SIAM Review*, 41(3):513–537, Sept. 1999.

[6] B. Tordoff and D.W. Murray. Guided sampling and consensus for motion estimation. In *Proc. ECCV*, 2002.

[7] P. H. S. Torr and A. Zisserman. MLESAC: A new robust estimator with application to estimating image geometry. *CVIU*, 78:138–156, 2000.

[8] N. Ueda and R. Nakano. Deterministic Annealing EM algorithm. *Neural Networks*, 11(2):271–282, 1998.

[9] N. Ueda, R. Nakano, Z. Ghahramani, and G. E. Hinton. *SMEM* algorithm for mixture models. *Neural Computation*, 12(9):2109–2128, 2000.

[10] G. Wei and M. Tanner. A Monte Carlo implementation of the EM algorithm. *Journal American Statistical Society*, 85:699–704, 1990.
